# Patterns of damage in neural networks: The effects of lesion area, shape and number

**Eytan Ruppin and James A. Reggia** *
Department of Computer Science
A.V. Williams Bldg.
University of Maryland
College Park, MD 20742
ruppin@cs.umd.edu   reggia@cs.umd.edu

## Abstract

Current understanding of the effects of damage on neural networks is rudimentary, even though such understanding could lead to important insights concerning neurological and psychiatric disorders. Motivated by this consideration, we present a simple analytical framework for estimating the functional damage resulting from focal structural lesions to a neural network. The effects of focal lesions of varying area, shape and number on the retrieval capacities of a spatially-organized associative memory. Although our analytical results are based on some approximations, they correspond well with simulation results. This study sheds light on some important features characterizing the clinical manifestations of multi-infarct dementia, including the strong association between the number of infarcts and the prevalence of dementia after stroke, and the 'multiplicative' interaction that has been postulated to occur between Alzheimer's disease and multi-infarct dementia.

*Dr. Reggia is also with the Department of Neurology and the Institute of Advanced Computer Studies at the University of Maryland.

# 1   Introduction

Understanding the response of neural nets to structural/functional damage is important for a variety of reasons, e.g., in assessing the performance of neural network hardware, and in gaining understanding of the mechanisms underlying neurological and psychiatric disorders. Recently, there has been a growing interest in constructing neural models to study how specific pathological neuroanatomical and neurophysiological changes can result in various clinical manifestations, and to investigate the functional organization of the symptoms that result from specific brain pathologies (reviewed in [1, 2]). In the area of associative memory models specifically, early studies found an increase in memory impairment with increasing lesion severity (in accordance with Lashley's classical 'mass action' principle), and showed that slowly developing lesions have less pronounced effects than equivalent acute lesions [3]. Recently, it was shown that the gradual pattern of clinical deterioration manifested in the majority of Alzheimer's patients can be accounted for, and that different synaptic compensation rates can account for the observed variation in the severity and progression rate of this disease [4]. However, this past work is limited in that model elements have no spatial relationships to one another (all elements are conceptually equidistant). Thus, as there is no way to represent focal (localized) damage in such networks, it has not been possible to study the functional effect of focal lesions and to compare them with that caused by diffuse lesions.

The limitations of past work led us to use spatially-organized neural network for studying the effects of different types of lesions (we use the term lesion to mean any type of structural and functional damage inflicted on an initially intact neural network). The elements in our model, which can be thought of as representing neurons, or micro-columnar units, form a 2-dimensional array (whose edges are connected, forming a torus to eliminate edge effects), and each unit is connected primarily to its nearby neighbors, as is the case in the cortex [5]. It has recently been shown that such spatially-organized attractor networks can function reasonably well as associative memory devices [6]. This paper presents the first detailed analysis of the effects of lesions of various size, form and number on the memory performance of spatially-organized attractor neural networks. Assuming that these networks are a plausible model of some frontal and associative cortical areas (see, e.g., [7]), our results shed light on the clinical progress of disorders such as stroke and dementia.

In the next section, we derive a theoretical framework that characterizes the effects of focal lesions on an associative network's performance. This framework, which is formulated in very general terms, is then examined via simulations in Section 3, which show a remarkable quantitative fit with the theoretical predictions, and are compared with simulations examining performance with diffuse damage. Finally, the clinical significance of our results is discussed in Section 4.

# 2   Analyzing the effects of focal lesions

Our analysis pertains to the case where in the pre-damaged network, all units have an approximately similar average level of activity [1]. A focal *structural lesion*

(anatomical lesion), denoting the area of damage and neuronal death, is modeled by clamping the activity of the lesioned units to zero. As a result of this primary lesion, the activity of surrounding units may be decreased, resulting in a secondary area of *functional lesion*, as illustrated in Figure 1. We are primarily interested in large focal lesions, where the area $s$ of the lesion is significantly greater than the local neighborhood region from which each unit receives its inputs. Throughout our analysis we shall hold the working assumption that, traversing from the border of the lesion outwards, the activity of units gradually rises from zero until it reaches its normal, predamaged levels, at some distance $d$ from the lesion's border (see Figure 1). As $s$ is large and $d$ is determined by local interactions on the borders of the structural lesion, we may reasonably assume that the value of $d$ is independent of the lesion size, and depends primarily on the specific network characteristics, such as it architecture, dynamics, and memory load.

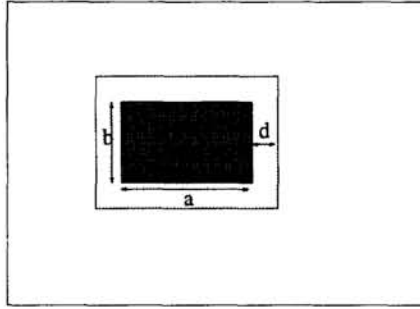

Figure 1: A sketch of a structural (dark shading) and surrounding functional (light shading) rectangular lesion.

Let the intact baseline performance level of the network be denoted as $P(0)$, and let the network size be $A$. The network's performance denotes how accurately it retrieves the correct memorized patterns given a set of input cues, and is defined formally below. A structural lesion of area $s$ (dark shading in Figure 1), causing a functional lesion of area $\Delta_s$ (light shading in Figure 1), will then result in a performance level of approximately

$$P(s) = \frac{P(0)\,[A - (s + \Delta_s)] + P_\Delta \Delta_s}{A - s} = P(0) - (\Delta P \Delta_s)/(A - s) , \qquad (1)$$

where $P_\Delta$ denotes the average level of performance over $\Delta_s$, and $\Delta P = P(0) - P_\Delta$. $P(s)$ hence reflects the performance level over the remaining viable parts of the network, discarding the structurally damaged region. Bearing these definitions in mind, a simple analysis shows that the effect of focal lesions is governed by the following rules.

Consider a symmetric, circular structural lesion of size $s = \pi r^2$. $\Delta_s$, the area of functional damage following such a lesion is then (assuming large lesions and hence $\sqrt{s} > d$)

*Rule 1:*

$$\Delta_s \cong \sqrt{4\pi} d \sqrt{s} , \qquad (2)$$

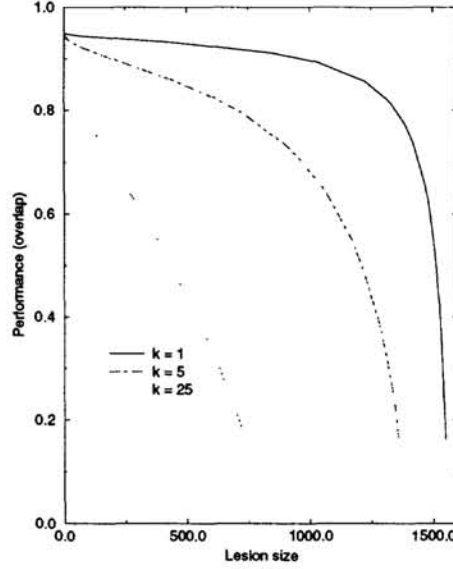

Figure 2: Theoretically predicted network performance as a function of a single focal structural lesion's size (area): analytic curves obtained for different $k$ values; $A = 1600$.

and

$$P(s) \cong P(0) - \frac{k\sqrt{s}}{A - s} \; , \tag{3}$$

for some constant $k = \sqrt{4\pi}d\Delta P$. Thus, the area of functional damage surrounding a single focal structural lesion is proportional to the square root of the structural lesion's area. Some analytic performance/lesioning curves (for various $k$ values) are illustrated in Figure 2. Note the different qualitative shape of these curves as a function of $k$. Letting $x = s/A$ be the *fraction of structural damage*, we have

$$P(x) \cong P(0) - \frac{k\sqrt{x}}{1 - x}\frac{1}{\sqrt{A}} \; , \tag{4}$$

that is, the same *fraction* $x$ of damage results in less performance decrease in larger networks. This surprising result testifies to the possible protective value of having functional 'modular' cortical networks of large size.

Expressions 3 and 4 are valid also when the structural lesion has a square shape. To study the effect of the structural lesion's shape, we consider the area $\Delta_{s[n]}$ of a functional lesion resulting from a rectangular focal lesion of size $s = a \cdot b$ (see Figure 1), where, without loss of generality, $n = a/b \geq 1$. Then, for large $n$, we find that the functional damage of a rectangular structural lesion of fixed size increases as its shape is more elongated, following
*Rule 2*:

$$\Delta_{s[n]} \cong \sqrt{n/4}\Delta_s \; , \tag{5}$$

and

$$P(s) \cong P(0) - \frac{k\sqrt{ns}}{2(A - s)} \; . \tag{6}$$

To study the effect of the number of lesions, consider the area $\Delta_s{}^m$ of a functional lesion composed of $m$ focal rectangular structural lesions (with sides $a = n \cdot b$), each of area $s/m$. We find that the functional damage increases with the number of focal sub-lesions (while total structural lesion area is held constant), according to *Rule 3*:

$$\Delta_s{}^m \geq \sqrt{m}\Delta_{s[n]} \,, \tag{7}$$

and

$$P(s) \cong P(0) - \frac{k\sqrt{mns}}{2(A-s)} \,. \tag{8}$$

While Rule 3 presents a lower bound on the functional damage which may actually be significantly larger and involves no approximations, Rule 2 presents an upper bound on the actual functional damage. As we shall show in the next section, the number of lesions actually affects the network performance significantly more than its precise shape.

## 3   Numerical Simulation Results

We now turn to examine the effect of lesions on the performance of an associative memory network via simulations. The goal of these simulations is twofold. First, to examine how accurately the general but approximate theoretical results presented above describe the actual performance degradation in a specific associative network. Second, to compare the effects of focal lesions to those of diffuse ones, as the effect of diffuse damage cannot be described as a limiting case within the framework of our analysis. Our simulations were performed using a standard Tsodyks-Feigelman attractor neural network [8]. This is a Hopfield-like network which has several features which make it more biologically plausible [4], such as low activity and non-zero positive thresholds. In all the experiments, 20 sparse random $\{0, 1\}$ memory patterns (with a fraction of $p \ll 1$ of 1's) were stored in a network of $N = 1600$ units, placed on a 2-dimensional lattice. The network has spatially organized connectivity, where each unit has 60 incoming connections determined randomly with a Gaussian probability $\phi(z) = \sqrt{1/2\pi}\exp(-z^2/2\sigma^2)$, where $z$ is the distance between two units in the array. When $\sigma$ is small, each unit is connected primarily to its nearby neighbors. As in [4], the cue input patterns are presented via an external field of magnitude $e = 0.035$, and the noise level is $T = 0.005$. The performance of the network is measured (over the viable, non-lesioned units) by the standard *overlap* measure which denotes the similarity between the final state $S$ the network converges to and the memory pattern $\xi^\mu$ that was cued in that trial, defined by

$$m^\mu(t) = \frac{1}{p(1-p)N} \sum_{i=1}^{N} (\xi_i^\mu - p)S_i(t) \,. \tag{9}$$

In all simulations we report the average overlap achieved over 100 trials.

We first studied the network's performance at various $\sigma$ values. Figure 3a displays how the performance of the network degrades when *diffuse* structural lesions of increasing size are inflicted upon it (i.e., randomly selected units are clamped to zero), while Figure 3b plots the performance as a function of the size of a single square-shaped focal lesion. As is evident, spatially-organized connectivity enables

the network to maintain its memory retrieval capacities in face of focal lesions of considerable size. Diffuse lesions are always more detrimental than *single* focal lesions of identical size. Also plotted in Figure 3b is the analytical curve calculated via expression (3) (with $k = 5$), which shows a nice fit with the actual performance of the spatially-connected network parametrized by $\sigma = 1$. Concentrating on the study of focal lesions in a spatially-connected network, we adhere to the values $\sigma = 1$ and $k = 5$ hereafter, and compare the analytical and numerical results. With these values, the analytical curves describing the performance of the network as a function of the fraction of the network lesioned (obtained using expression 4) are similar to the corresponding numerical results.

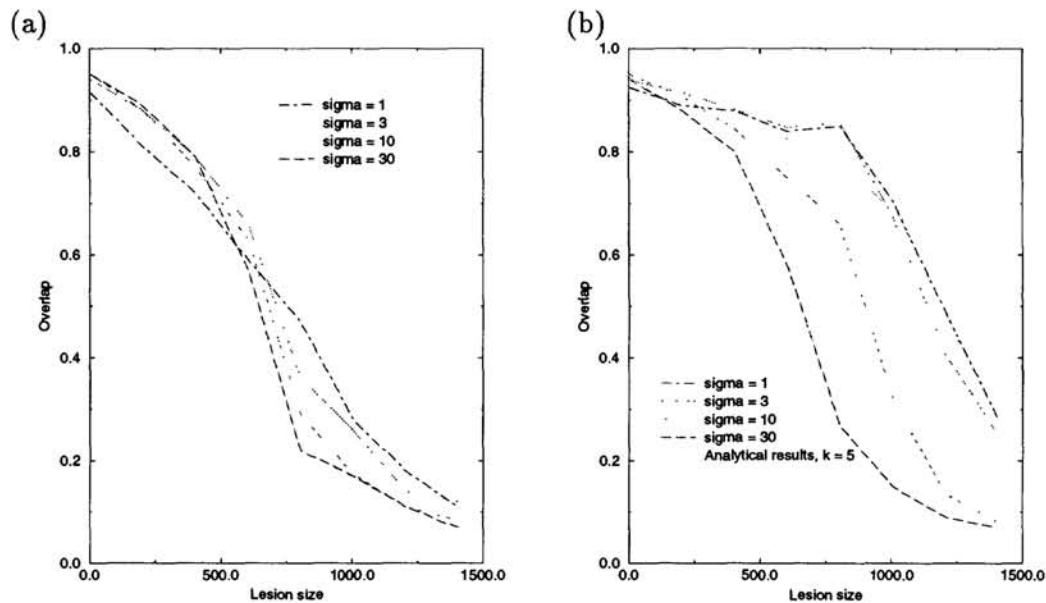

Figure 3: Network performance as a function of lesion size: simulation results obtained in four different networks, each characterized by a distinct distribution of spatially-organized connectivity. (a) Diffuse lesions. (b) Focal lesions.

To examine Rule 2, a rectangular structural lesion of area $s = 300$ was induced in the network. As shown in Figure 4a, as the ratio $n$ between the sides is increased, the network's performance further decreases, but this effect is relatively mild. The markedly stronger effect of varying the lesion number (described by Rule 3) is demonstrated in figure 4b, which shows the effect of multiple lesions composed of $2, 4, 8$ and $16$ separate focal lesions. For comparison, the performance achieved with a diffuse lesion of similar size is plotted on the $20'th$ x-ordinate. It is interesting to note that a sufficiently large multiple focal lesion ($s = 512$) can cause a larger performance decrease than a diffuse lesion of similar size. That is, at some point, when the size of each individual focal lesion becomes small in relation to the spread of each unit's connectivity, our analysis looses its validity, and Rule 3 ceases to hold.

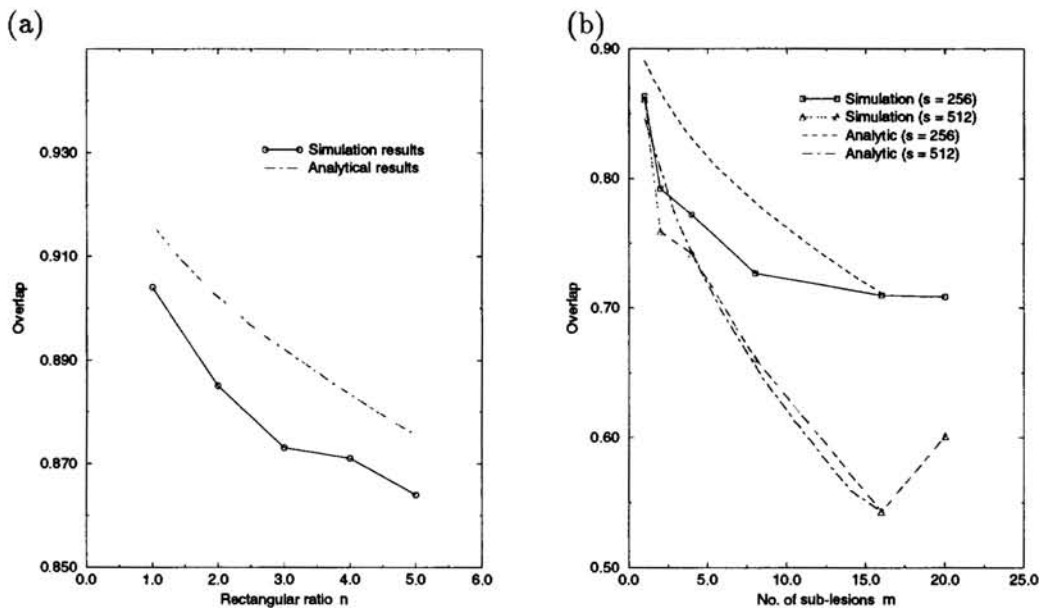

Figure 4: Network performance as a function of focal lesion shape (a) and number (b). Both numerical and analytical results are displayed. In Figure 4b, the x-ordinate denotes the number of separate sub-lesions (1,2,4,8,16), and, for comparison, the performance achieved with a diffuse lesion of similar size is plotted on the $20'th$ x-ordinate.

## 4  Discussion

We have presented a simple analytical framework for studying the effects of focal lesions on the functioning of spatially organized neural networks. The analysis presented is quite general and a similar approach could be adopted to investigate the effect of focal lesions in other neural models. Using this analysis, specific scaling rules have been formulated describing the functional effects of structural focal lesions on memory retrieval performance in associative attractor networks. The functional lesion scales as the square root of the size of a single structural lesion, and the form of the resulting performance curve depends on the impairment span $d$. Surprisingly, the same fraction of damage results in significantly less performance decrease in larger networks, pointing to their relative robustness. As to the effects of shape and number, elongated structural lesions cause more damage than more symmetrical ones. However, the number of sub-lesions is the most critical factor determining the functional damage and performance decrease in the model. Numerical studies show that in some conditions multiple lesions can damage performance more than diffuse damage, even though the amount of lost innervation is always less in a multiple focal lesion than with diffuse damage.

Beyond its computational interest, the study of the effects of focal damage on the performance of neural network models can lead to a better understanding of functional impairments accompanying focal brain lesions. In particular, we are interested in *multi-infarct dementia*, a frequent cause of dementia (chronic deterioration of cognitive and memory capacities) characterized by a series of multiple, aggregat-

ing focal lesions. Our results indicate a significant role for the number of infarcts in determining the extent of functional damage and dementia in multi-infarct disease. In our model, multiple focal lesions cause a much larger deficit than their simple 'sum', i.e., a single lesion of equivalent total size. This is consistent with clinical studies that have suggested the main factors related to the prevalence of dementia after stroke to be the infarct number and site, and not the overall infarct size, which is related to the prevalence of dementia in a significantly weaker manner [9, 10]. Our model also offers a possible explanation to the 'multiplicative' interaction that has been postulated to occur between co-existing Alzheimer and multi-infarct dementia [10], and to the role of cortical atrophy in increasing the prevalence of dementia after stroke; in accordance with our model, it is hypothesized that atrophic degenerative changes will lead to an increase in the value of $d$ (and hence of $k$) and increase the functional damage caused by a lesion of given structural size. This hypothesis, together with a detailed study of the effects of the various network parameters on the value of $d$, are currently under further investigation.

**Acknowledgements**
This research has been supported by a Rothschild Fellowship to Dr. Ruppin and by Awards NS29414 and NS16332 from NINDS.

## Footnotes

[1]This is true in general for associative memory networks, when the activity of each unit is averaged over a sufficiently long time span.

# References

[1] J. Reggia, R. Berndt, and L. D'Autrechy. Connectionist models in neuropsychology. In *Handbook of Neuropsychology*, volume 9. 1994, in press.

[2] E. Ruppin. Neural modeling of psychiatric disorders. *Network: Computation in Neural Systems*, 1995. Invited review paper, to appear.

[3] J.A. Anderson. Cognitive and psychological computation with neural models. *IEEE Trans. on Systems,Man, and Cybernetics*, SMC-13(5):799–815, 1983.

[4] D. Horn, E. Ruppin, M. Usher, and M. Herrmann. Neural network modeling of memory deterioration in alzheimer's disease. *Neural Computation*, 5:736–749, 1993.

[5] A. M. Thomson and J. Deuchars. Temporal and spatial properties of local circuits in the neocortex. *Trends in neuroscience*, 17(3):119–126, 1994.

[6] J.M. Karlholm. Associative memories with short-range higher order couplings. *Neural Networks*, 6:409–421, 1993.

[7] F.A.W. Wilson, S.P.O Scalaidhe, and P.S. Goldman-Rakic. Dissociation of object and spatial processing domains in primate prefrontal cortex. *Science*, 260:1955–1958, 1993.

[8] M.V. Tsodyks and M.V. Feigel'man. The enhanced storage capacity in neural networks with low activity level. *Europhys. Lett.*, 6:101 – 105, 1988.

[9] T.K. Tatemichi, M.A. Foulkes, J.P. Mohr, J.R. Hewitt, D. B. Hier, T.R. Price, and P.A. Wolf. Dementia in stroke survivors in the stroke data bank cohort. *Stroke*, 21:858–866, 1990.

[10] T. K. Tatemichi. How acute brain failure becomes chronic: a view of the mechanisms of dementia related to stroke. *Neurology*, 40:1652–1659, 1990.